# A High Performance *k*-NN Classifier Using a Binary Correlation Matrix Memory

**Ping Zhou**
zhoup@cs.york.ac.uk

**Jim Austin**
austin@cs.york.ac.uk

**John Kennedy**
johnk@cs.york.ac.uk

Advanced Computer Architecture Group
Department of Computer Science
University of York, York YO10 5DD, UK

## Abstract

This paper presents a novel and fast *k*-NN classifier that is based on a binary CMM (Correlation Matrix Memory) neural network. A robust encoding method is developed to meet CMM input requirements. A hardware implementation of the CMM is described, which gives over 200 times the speed of a current mid-range workstation, and is scaleable to very large problems. When tested on several benchmarks and compared with a simple *k*-NN method, the CMM classifier gave less than 1% lower accuracy and over 4 and 12 times speed-up in software and hardware respectively.

## 1 INTRODUCTION

Pattern classification is one of most fundamental and important tasks, and a *k*-NN rule is applicable to a wide range of classification problems. As this method is too slow for many applications with large amounts of data, a great deal of effort has been put into speeding it up via complex pre-processing of training data, such as reducing training data (Dasarathy 1994) and improving computational efficiency (Grother & Candela 1997). This work investigates a novel *k*-NN classification method that uses a binary correlation matrix memory (CMM) neural network as a pattern store and match engine. Whereas most neural networks need a long iterative training time, a CMM is simple and quick to train. It requires only one-shot storage mechanism and simple binary operations (Willshaw & Buneman 1969), and it has highly flexible and fast pattern search ability. Therefore, the combination of CMM and *k*-NN techniques is likely to result in a generic and fast classifier. For most classification problems, patterns are in the form of multi-dimensional real numbers, and appropriate quantisation and encoding are needed to convert them into binary inputs to a CMM. A robust quantisation and encoding method is developed to meet requirements for CMM input codes, and to overcome the common problem of identical data points in many applications, e.g. background of images or normal features in a diagnostic problem.

Many research projects have applied the CMM successfully to commercial problems, e.g. symbolic reasoning in the AURA (Advanced Uncertain Reasoning Architecture) approach

(Austin 1996), chemical structure matching and post code matching. The execution of the CMM has been identified as the bottleneck. Motivated by the needs of these applications for a further high speed processing, the CMM has been implemented in dedicated hardware, i.e. the PRESENCE architecture. The primary aim is to improve the execution speed over conventional workstations in a cost-effective way.

The following sections discuss the CMM for pattern classification, describe the PRESENCE architecture (the hardware implementation of CMM), and present experimental results on several benchmarks.

## 2 BINARY CMM *k*-NN CLASSIFIER

The key idea (Figure 1) is to use a CMM to pre-select a small sub-set of training patterns from a large number of training data, and then to apply the *k*-NN rule to the sub-set. The CMM is fast but produces spurious errors as a side effect (Turner & Austin 1997); these are removed through the application of the *k*-NN rule. The architecture of the CMM classifier (Figure 1) includes an encoder (detailed in 2.2) for quantising numerical inputs and generating binary codes, a CMM pattern store and match engine and a conventional *k*-NN module as detailed below.

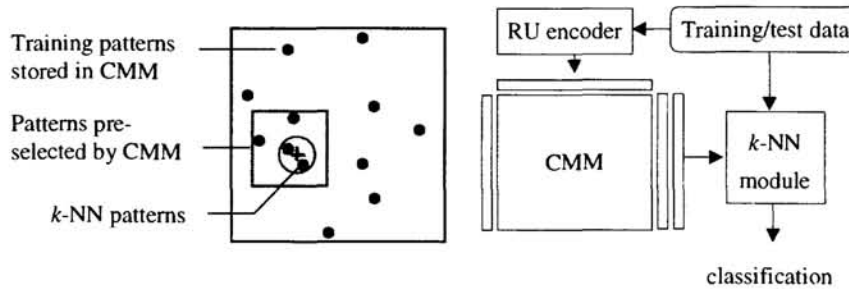

Figure 1: Architecture of the binary CMM *k*-NN classifier

### 2.1 PATTERN MATCH AND CLASSIFICATION WITH CMM

A correlation matrix memory is basically a single layer network with binary weights $M$. In the training process a unique binary vector or separator $s_i$ is generated to label an unseen input binary vector $p_i$; the CMM learns their association by performing the following logical ORing operation:

$$M = \bigvee_i s_i^T p_i \tag{1}$$

In a recall process, for a given test input vector $p_k$, the CMM performs:

$$v_k = M p_k^T = \left( \bigvee_i s_i^T p_i \right) p_k^T \tag{2}$$

followed by thresholding $v_k$ and recovering individual separators. For speed, it is appropriate to use a fixed thresholding method and the threshold is set to a level proportional to the number of '1' bits in the input pattern to allow an exact or partial match. To understand the recall properties of the CMM, consider the case where a known pattern $p_k$ is represented, then Equation 2 can be written as the following when two different patterns are orthogonal to each other:

$$v_k = s_k^T p_k p_k^T + \bigvee_{i \neq k} s_i^T p_i p_k^T = n_p s_k^T \tag{3}$$

where $n_p$ is a scalar, i.e. the number of '1' bits in $p_k$, and $p_i p_k^T = 0$ for $i \neq k$. Hence a perfect recall of $s_k$ can be obtained by thresholding $v_k$ at the level $n_p$. In practice 'partially orthogonal' codes may be used to increase the storage capacity of the CMM and the recall noise can be removed via appropriately thresholding $v_k$ (as $p_i p_k^T \leq n_p$ for $i \neq k$)

and post-processing (e.g. applying $k$-NN rule). Sparse codes are usually used, i.e. only a few bits in $s_k$ and $p_t$ being set to '1', as this maximises the number of codes and minimises the computation time (Turner & Austin 1997). These requirements for input codes are often met by an encoder as detailed below.

The CMM exhibits an interesting 'partial match' property when the data dimensionality $d$ is larger than one and input vector $p_i$ consists of $d$ concatenated components. If two different patterns have some common components, $v_k$ also contains separators for partially matched patterns, which can be obtained at lower threshold levels. This partial or near match property is useful for pattern classification as it allows the retrieval of stored patterns that are close to the test pattern in Hamming distance.

From those training patterns matched by the CMM engine, a test pattern is classified using the $k$-NN rule. Distances are computed in the original input space to minimise the information loss due to quantisation and noise in the above match process. As the number of matches returned by the CMM is much smaller than the number of training data, the distance computation and comparison are dramatically reduced compared with the simple $k$-NN method. Therefore, the speed of the classifier benefits from fast training and matching of the CMM, and the accuracy gains from the application of the $k$-NN rule for reducing information loss and noise in the encoding and match processes.

## 2.2 ROBUST UNIFORM ENCODING

Figure 2 shows three stages of the encoding process. $d$-dimensional real numbers, $x_i$, are quantised as $y_i$; sparse and orthogonal binary vectors, $c_i$, are generated and concatenated to form a CMM input vector.

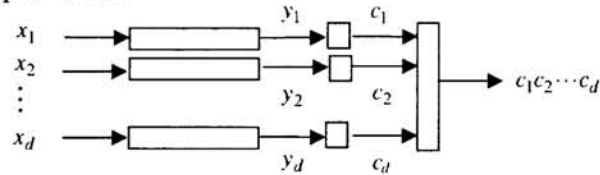

Figure 2: Quantisation, code generation and concatenation

CMM input codes should be distributed as uniformly as possible in order to avoid some parts of the CMM being used heavily while others are rarely used. The code uniformity is met at the quantisation stage. For a given set of $N$ training samples in some dimension (or axis), it is required to divide the axis into $N_b$ small intervals, called bins, such that they contain uniform numbers of data points. As the data often have a non-uniform distribution, the sizes of these bins should be different. It is also quite common for real world problems that many data points are identical. For instance, there are 11%-99.9% identical data in benchmarks used in this work. Our robust quantisation method described below is designed to cope with the above problems and to achieve a maximal uniformity.

In our method data points are first sorted in ascending order, $N_t$ identical points are then identified, and the number of non-identical data points in each bin is estimated as $N_p = (N - N_t)/N_b$. Bin boundaries or partitions are determined as follows. The right boundary of a bin is initially set to the next $N_p$-th data point in the ordered data sequence; the number of identical points on both sides of the boundary is identified; these are either included in the current or next bin. If the number of non-identical data points in the last bin is $N_l$ and $N_l \geq (N_p + N_b)$, $N_p$ may be increased by $(N_l - N_p)/N_b$ and the above partition process may be repeated to increase the uniformity. Boundaries of bins obtained become parameters of the encoder in Figure 2. In general it is appropriate to choose $N_b$ such that each bin contains a number of samples, which is larger than $k$ nearest neighbours for the optimal classification.

## 3 THE PRESENCE ARCHITECTURE

The pattern match and store engine of the CMM $k$-NN classifier has been implemented using a novel hardware based CMM architecture, i.e. the PRESENCE.

### 3.1 ARCHITECTURE DESIGN

Important design decisions include the use of cheap memory, and not embedding both the weight storage and the training and testing in hardware (VLSI). This arises because the applications commonly use CMMs with over 100Mb of weight memory, which would be difficult and expensive to implement in custom silicon. VME and PCI are chosen to host on industry standard buses and to allow widespread application.

The PRESENCE architecture implements the control logic and accumulators, i.e. the core of the CMM. As shown in Figure 3a binary input selects rows from the CMM that are added, thresholded using $L$-max (Austin & Stonham 1987) or fixed global thresholding, and then returned to the host for further processing. The PRESENCE architecture shown in Figure 3b consists of a bus interface, a buffer memory which allows interleaving of memory transfer and operation of the PRESENCE system, a SATCON and SATSUM combination that accumulates and thresholds the weights. The data bus connects to a pair of memory spaces, each of which contains a control block, an input block and an output block. Thus the PRESENCE card is a memory mapping device, that uses interrupts to confirm the completion of each operation. For efficiency, two memory input/output areas are provided to be acted on from the external bus and used by the card. The control memory input block feeds to the control unit, which is a FPGA device. The input data are fed to the weights and the memory area read is then passed to a block of accumulators. In our current implementation the data width of each FPGA device is 32 bits, which allows us to add a 32 bit row from the weights memory in one cycle per device

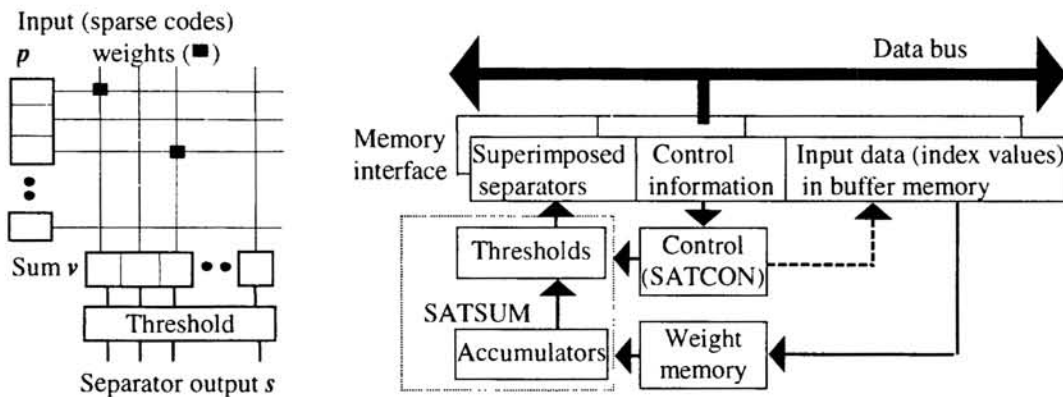

Figure 3: (a) correlation matrix memory, and (b) overall architecture of PRESENCE

Currently we have 16Mb of 25ns static memory implemented on the VME card, and 128 Mb of dynamic (60ns) memory on the PCI card. The accumulators are implemented along with the thresholding logic on another FPGA device (SATSUM). To enable the SATSUM processors to operate faster, a 5 stage pipeline architecture was used, and the data accumulation time is reduced from 175ns to 50ns. All PRESENCE operations are supported by a C++ library that is used in all AURA applications. The design of the SATCON allows many SATSUM devices to be used in parallel in a SIMD configuration. The VME implementation uses 4 devices per board giving a 128 bit wide data path. In addition the PCI version allows daisy chaining of cards allowing a 4 card set for a 512 bit wide data path. The complete VME card assembly is shown in Figure 4. The SATCON and SATSUM devices are mounted on a daughter board for simple upgrading and alteration. The weights memory, buffer memory and VME interface are held on the mother board.

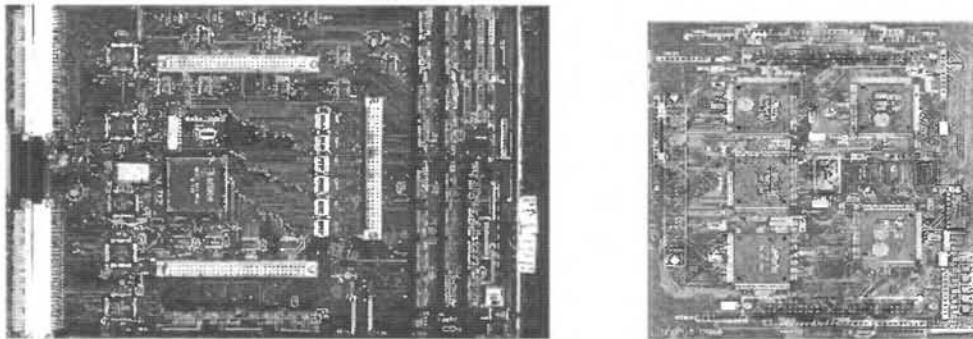

Figure 4: The VME based PRESENCE card (a) motherboard, and (b) daughterboard

## 3.2 PERFORMANCE

By an analysis of the state machines used in the SATCON device the time complexity of the approach can be calculated. Equation 4 is used to calculate the processing time, $T$, in seconds to recall the data with $N$ index values, a separator size of $S$, $R$ 32 bit SATSUM devices, and the clock period of $C$.

$$T = C\left[23 + \left((s-1)/32R + 1\right)(N + 38 + 2R)\right] \qquad (4)$$

A comparison with a Silicon Graphics 133MHz R4600SC Indy in Table 1 shows the speed up of the matrix operation (Equation 2) for our VME implementation (128 bits wide) using a fixed threshold. The values for processing rate are given in millions of binary weight additions per-second (MW/s). The system cycle time needed to sum a row of weights into the counters (i.e. time to accumulate one line) is 50ns for the VME version and 100ns for the PCI version. In the PCI form, we will use 4 closely coupled cards, which result in a speed-up of 432. The build cost of the VME card was half the cost of the baseline SGI Indy machine, when using 4Mb of 20ns static RAM. In the PCI version the cost is greatly reduced through the use of dynamic RAM devices allowing a 128Mb memory to be used for the same cost, allowing only a 2x slower system with 32x as much memory per card (note that 4 cards used in Table 1 hold 512Mb of memory).

Table 1: Relative speed-up of the PRESENCE architecture

| Platform | Processing Rate | Relative Speed |
|---|---|---|
| Workstation | 11.8 MW/s | 1 |
| 1 Card VME implementation | 2557MW/s | 216 |
| Four card PCI system (estimate) | 17,114MW/s | 432 |

The training and recognition speed of the system are approximately equal. This is particularly useful in on-line applications, where the system must learn to solve the problem incrementally as it is presented. In particular, the use of the system for high speed reasoning allows the rules in the system to be altered without the long training times of other systems. Furthermore our use of the system for a $k$-NN classifier also allows high speed operation compared with a conventional implementation of the classifier, while still allowing very fast training times.

## 4 RESULTS ON BENCHMARKS

Performance of the robust quantisation method and the CMM classifier have been evaluated on four benchmarks consisting of large sets of real world problems from the Statlog project (Michie & Spiegelhalter 1994), including a satellite image database, letter image recognition database, shuttle data set and image segmentation data set. To visualise the result of quantisation, Figure 5a shows the distribution of numbers of data points of the $8^{th}$ feature of the image segment data for equal-size bins. The distribution represents

the inherent characteristics of the data. Figure 5b shows our robust quantisation (RQ) has resulted in the uniform distribution desired.

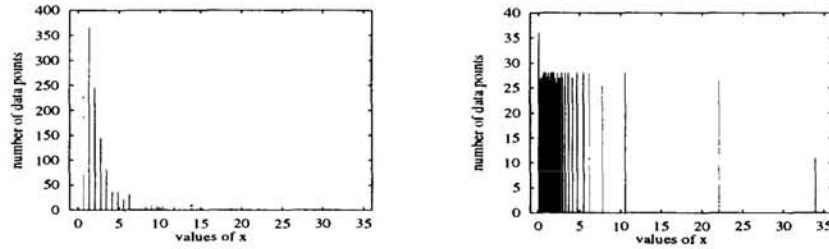

Figure 5: Distributions of the image segment data for (a) equal bins, (b) RQ bins

We compared the CMM classifier with the simple $k$-NN method, multi-layer perceptron (MLP) and radial basis function (RBF) networks (Zhou and Austin 1997). In the evaluation we used the CMM software libraries developed in the project AURA at the University of York. Between 1 and 3 '1' bits are set in input vectors and separators. Experiments were conducted to study influences of a CMM's size on classification rate (c-rate) on test data sets and speed-up measured against the $k$-NN method (as shown in Figure 6). The speed-up of the CMM classifier includes the encoding, training and test time. The effects of the number of bins $N_b$ on the performance were also studied.

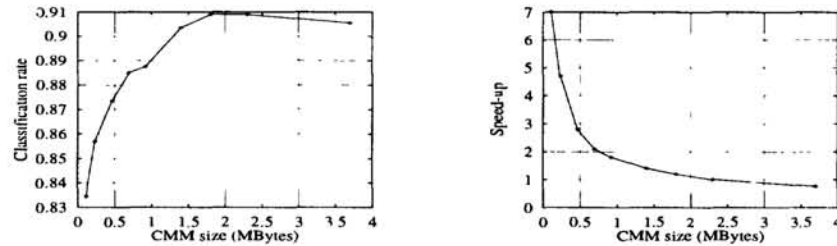

Figure 6: Effects of the CMM size on (a) c-rate and (b) speed-up on the satellite image data

Choices of the CMM size and the number of bins may be application dependent, for instance, in favour of the speed or accuracy. In the experiment it was required that the speed-up is not 4 times less and c-rate is not 1% lower than that of the $k$-NN method. Table 2 contains the speed-up of MLP and RBF networks and the CMM on the four benchmarks. It is interesting to note that the $k$-NN method needed no training. The recall of MLP and RBF networks was very faster but their training was much slower than that of the CMM classifier. The recall speed-up of the CMM was 6-23 times, and the overall speed-up (including training and recall time) was 4-15x. When using the PRESENCE, i.e. the dedicated CMM hardware, the speed of the CMM was further increased over 3 times. This is much less than the speed-up of 216 given in Table 1 because of recovering separators and $k$-NN classification are performed in software.

Table 2: Speed-up of MLP, RBF and CMM relative to the simple $k$-NN method

|  | Image segment | | Satellite image | | Letter | | Shuttle | |
|---|---|---|---|---|---|---|---|---|
| method | training | test | training | Test | training | test | training | test |
| MLPN | 0.04 | 18 | 0.2 | 28.4 | 0.2 | 96.5 | 4.2 | 587.2 |
| RBFN | 0.09 | 9 | 0.07 | 20.3 | 0.3 | 66.4 | 1.8 | 469.7 |
| simple $k$-NN | - | 1 | - | 1 | - | 1 | - | 1 |
| CMM | 18 | 9 | 15.8 | 5.7 | 24.6 | 6.8 | 43 | 23 |

The classification rates by the four methods are given in Table 3, which shows the CMM classifier performed only 0-1% less accurate than the $k$-NN method.

Table 3: Classification rates of four methods on four benchmarks

|  | Image segment | Satellite image | Letter | Shuttle |
|---|---|---|---|---|
| MLPN | 0.950 | 0.914 | 0.923 | 0.998 |
| RBFN | 0.939 | 0.914 | 0.941 | 0.997 |
| simple $k$-NN | 0.956 | 0.906 | 0.954 | 0.999 |
| CMM | 0.948 | 0.901 | 0.945 | 0.999 |

## 5 CONCLUSIONS

A novel classifier is presented, which uses a binary CMM for storing and matching a large amount of patterns efficiently, and the $k$-NN rule for classification. The RU encoder converts numerical inputs into binary ones with the maximally achievable uniformity to meet requirements of the CMM. Experimental results on the four benchmarks show that the CMM classifier, compared with the simple $k$-NN method, gave slightly lower classification accuracy (less than 1% lower) and over 4 times speed in software and 12 times speed in hardware. Therefore our method has resulted in a generic and fast classifier.

This paper has also described a hardware implementation of a FPGA based chip set and a processor card that will support the execution of binary CMM. It has shown the viability of using a simple binary neural network to achieve high processing rates. The approach allows both recognition and training to be achieved at speeds well above two orders of magnitude faster than conventional workstations at a much lower cost than the workstation. The system is scaleable to very large problems with very large weight arrays. Current research is aimed at showing that the system is scaleable, evaluating methods for the acceleration of the pre- and post processing tasks and considering greater integration of the elements of the processor through VLSI. For more details of the AURA project and the hardware described in this paper see http://www.cs.york.ac.uk/arch/nn/aura.html.

### Acknowledgements

We acknowledge British Aerospace and the Engineering and Physical Sciences Research Council (grant no. GR/K 41090 and GR/L 74651) for sponsoring the research. Our thanks are given to R Pack, A Moulds, Z Ulanowski, R Jennison and K Lees for their support.

### References

Willshaw, D.J., Buneman, O.P. & Longuet-Higgins, H.C. (1969) Non-holographic associative memory. Nature, Vol. 222, p960-962.

Austin, J. (1996) AURA, A distributed associative memory for high speed symbolic reasoning. In: Ron Sun (ed), Connectionist Symbolic Integration. Kluwer.

Turner, M. & Austin, J. (1997) Matching performance of binary correlation matrix memories. Neural Networks; 10:1637-1648.

Dasarathy, B.V. (1994) Minimal consistent set (MCS) identification for optimal nearest neighbor decision system design. IEEE Trans. Systems Man Cybernet; 24:511-517.

Grother, P.J., Candela, G.T. & Blue, J.L. (1997) Fast implementations of nearest neighbor classifiers. Pattern Recognition; 30:459-465.

Austin, J., Stonham, T.J. (1987) An associative memory for use in image recognition and occlusion analysis. Image and Vision Computing; 5:251-261.

Michie, D., Spiegelhalter, D.J. & Taylor, C.C. (1994) Machine learning, neural and statistical classification (Chapter 9). New York, Ellis Horwood.

Zhou, P. & Austin J. (1998) Learning criteria for training neural network classifiers. Neural Computing and Applications Forum; 7:334-342.
